# Computing Time Lower Bounds for Recurrent Sigmoidal Neural Networks

**Michael Schmitt**
Lehrstuhl Mathematik und Informatik, Fakultät für Mathematik
Ruhr-Universität Bochum, D–44780 Bochum, Germany
*mschmitt@lmi.ruhr-uni-bochum.de*

## Abstract

Recurrent neural networks of analog units are computers for real-valued functions. We study the time complexity of real computation in general recurrent neural networks. These have sigmoidal, linear, and product units of unlimited order as nodes and no restrictions on the weights. For networks operating in discrete time, we exhibit a family of functions with arbitrarily high complexity, and we derive almost tight bounds on the time required to compute these functions. Thus, evidence is given of the computational limitations that time-bounded analog recurrent neural networks are subject to.

## 1   Introduction

Analog recurrent neural networks are known to have computational capabilities that exceed those of classical Turing machines (see, e.g., Siegelmann and Sontag, 1995; Kilian and Siegelmann, 1996; Siegelmann, 1999). Very little, however, is known about their limitations. Among the rare results in this direction, for instance, is the one of Šíma and Orponen (2001) showing that continuous-time Hopfield networks may require exponential time before converging to a stable state. This bound, however, is expressed in terms of the size of the network and, hence, does not apply to fixed-size networks with a given number of nodes. Other bounds on the computational power of analog recurrent networks have been established by Maass and Orponen (1998) and Maass and Sontag (1999). They show that discrete-time recurrent neural networks recognize only a subset of the regular languages in the presence of noise. This model of computation in recurrent networks, however, receives its inputs as sequences. Therefore, computing time is not an issue since the network halts when the input sequence terminates. Analog recurrent neural networks, however, can also be run as "real" computers that get as input a vector of real numbers and, after computing for a while, yield a real output value. No results are available thus far regarding the time complexity of analog recurrent neural networks with given size.

We investigate here the time complexity of discrete-time recurrent neural networks that compute functions over the reals. As network nodes we allow sigmoidal units, linear units, and product units—that is, monomials where the exponents are ad-

justable weights (Durbin and Rumelhart, 1989). We study the complexity of real computation in the sense of Blum et al. (1998). That means, we consider real numbers as entities that are represented exactly and processed without restricting their precision. Moreover, we do not assume that the information content of the network weights is bounded (as done, e.g., in the works of Balcázar et al., 1997; Gavaldà and Siegelmann, 1999). With such a general type of network, the question arises which functions can be computed with a given number of nodes and a limited amount of time. In the following, we exhibit a family of real-valued functions $f_l, l \geq 1$, in one variable that is computed by some fixed size network in time $O(l)$. Our main result is, then, showing that every recurrent neural network computing the functions $f_l$ requires at least time $\Omega(l^{1/4})$. Thus, we obtain almost tight time bounds for real computation in recurrent neural networks.

## 2 Analog Computation in Recurrent Neural Networks

We study a very comprehensive type of discrete-time recurrent neural network that we call *general recurrent neural network* (see Figure 1). For every $k, n \in \mathbb{N}$ there is a recurrent neural architecture consisting of $k$ *computation nodes* $y_1, \ldots, y_k$ and $n$ *input nodes* $x_1, \ldots, x_n$. The *size* of a network is defined to be the number of its computation nodes. The computation nodes form a fully connected recurrent network. Every computation node also receives connections from every input node. The input nodes play the role of the input variables of the system. All connections are parameterized by real-valued adjustable *weights*. There are three types of computation nodes: product units, sigmoidal units, and linear units. Assume that computation node $i$ has connections from computation nodes weighted by $w_{i1}, \ldots, w_{ik}$ and from input nodes weighted by $v_{i1}, \ldots, v_{in}$. Let $y_1(t), \ldots, y_k(t)$ and $x_1(t), \ldots, x_n(t)$ be the values of the computation nodes and input nodes at time $t$, respectively. If node $i$ is a *product unit*, it computes at time $t + 1$ the value

$$y_i(t+1) = y_1^{w_{i1}}(t) \cdots y_k^{w_{ik}}(t) x_1^{v_{i1}}(t) \cdots x_n^{v_{in}}(t), \tag{1}$$

that is, after weighting them exponentially, the incoming values are multiplied. Sigmoidal and linear units have an additional parameter associated with them, the threshold or bias $\theta_i$. A *sigmoidal unit* computes the value

$$y_i(t+1) = \sigma(w_{i1}y_1(t) + \cdots + w_{ik}y_k(t) + v_{i1}x_1(t) + \cdots + v_{in}x_n(t) - \theta_i), \tag{2}$$

where $\sigma$ is the standard sigmoid $\sigma(z) = 1/(1 + e^{-z})$. If node $i$ is a *linear unit*, it simply outputs the weighted sum

$$y_i(t+1) = w_{i1}y_1(t) + \cdots + w_{ik}y_k(t) + v_{i1}x_1(t) + \cdots + v_{in}x_n(t) - \theta_i. \tag{3}$$

We allow the networks to be heterogeneous, that is, they may contain all three types of computation nodes simultaneously. Thus, this model encompasses a wide class of network types considered in research and applications. For instance, architectures have been proposed that include a second layer of linear computation nodes which have no recurrent connections to computation nodes but serve as output nodes (see, e.g., Koiran and Sontag, 1998; Haykin, 1999; Siegelmann, 1999). It is clear that in the definition given here, the linear units can function as these output nodes if the weights of the outgoing connections are set to 0. Also very common is the use of sigmoidal units with higher-order as computation nodes in recurrent networks (see, e.g., Omlin and Giles, 1996; Gavaldà and Siegelmann, 1999; Carrasco et al., 2000). Obviously, the model here includes these higher-order networks as a special case since the computation of a higher-order sigmoidal unit can be simulated by first computing the higher-order terms using product units and then passing their

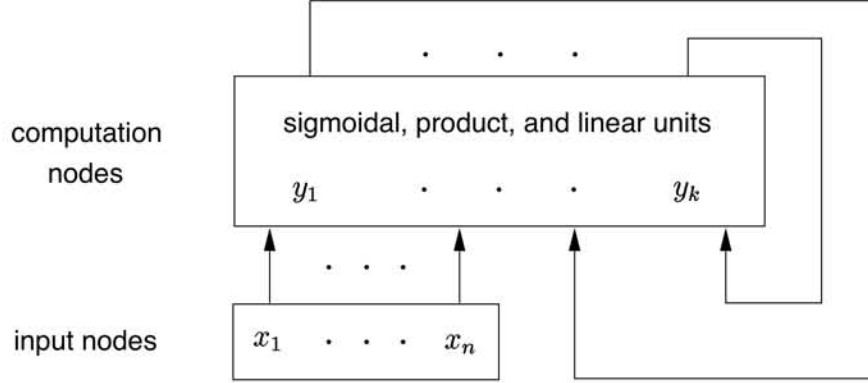

Figure 1: A general recurrent neural network of size $k$. Any computation node may serve as output node.

outputs to a sigmoidal unit. Product units, however, are even more powerful than higher-order terms since they allow to perform division operations using negative weights. Moreover, if a negative input value is weighted by a non-integer weight, the output of a product unit may be a complex number. We shall ensure here that all computations are real-valued. Since we are mainly interested in lower bounds, however, these bounds obviously remain valid if the computations of the networks are extended to the complex domain.

We now define what it means that a recurrent neural network $\mathcal{N}$ computes a function $f : \mathbb{R}^n \to \mathbb{R}$. Assume that $\mathcal{N}$ has $n$ input nodes and let $x \in \mathbb{R}^n$. Given $t \in \mathbb{N}$, we say that $\mathcal{N}$ *computes* $f(x)$ *in* $t$ *steps* if after initializing at time 0 the input nodes with $x$ and the computation nodes with some fixed values, and performing $t$ computation steps as defined in Equations (1), (2), and (3), one of the computation nodes yields the value $f(x)$. We assume that the input nodes remain unchanged during the computation. We further say that $\mathcal{N}$ *computes* $f$ *in time* $t$ if for every $x \in \mathbb{R}^n$, network $\mathcal{N}$ computes $f$ in at most $t$ steps. Note that $t$ may depend on $f$ but must be independent of the input vector. We emphasize that this is a very general definition of analog computation in recurrent neural networks. In particular, we do not specify any definite output node but allow the output to occur at any node. Moreover, it is not even required that the network reaches a stable state, as with attractor or Hopfield networks. It is sufficient that the output value appears at some point of the trajectory the network performs. A similar view of computation in recurrent networks is captured in a model proposed by Maass et al. (2001). Clearly, the lower bounds remain valid for more restrictive definitions of analog computation that require output nodes or stable states. Moreover, they hold for architectures that have no input nodes but receive their inputs as initial values of the computation nodes. Thus, the bounds serve as lower bounds also for the transition times between real-valued states of discrete-time dynamical systems comprising the networks considered here.

Our main tool of investigation is the Vapnik-Chervonenkis dimension of neural networks. It is defined as follows (see also Anthony and Bartlett, 1999): A *dichotomy* of a set $S \subseteq \mathbb{R}^n$ is a partition of $S$ into two disjoint subsets $(S_0, S_1)$ satisfying $S_0 \cup S_1 = S$. A class $\mathcal{F}$ of functions mapping $\mathbb{R}^n$ to $\{0, 1\}$ is said to *shatter* $S$ if for every dichotomy $(S_0, S_1)$ of $S$ there is some $f \in \mathcal{F}$ that satisfies $f(S_0) \subseteq \{0\}$ and $f(S_1) \subseteq \{1\}$. The *Vapnik-Chervonenkis (VC) dimension* of $\mathcal{F}$ is defined as

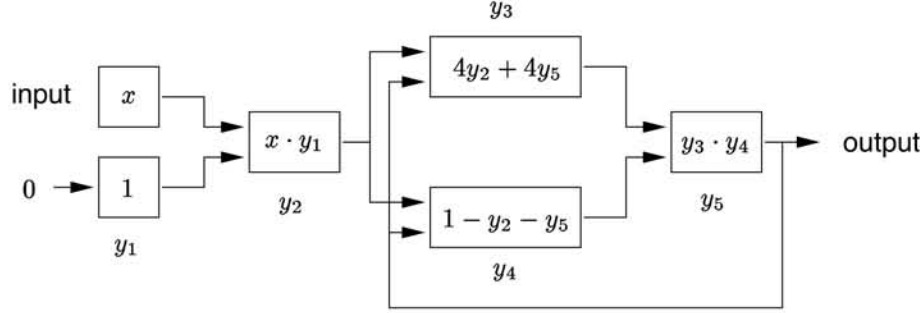

Figure 2: A recurrent neural network computing the functions $f_l$ in time $2l + 1$.

the largest number $m$ such that there is a set of $m$ elements shattered by $\mathcal{F}$. A neural network given in terms of an architecture represents a class of functions obtained by assigning real numbers to all its adjustable parameters, that is, weights and thresholds or a subset thereof. The output of the network is assumed to be thresholded at some fixed constant so that the output values are binary. The VC dimension of a neural network is then defined as the VC dimension of the class of functions computed by this network.

In deriving lower bounds in the next section, we make use of the following result on networks with product and sigmoidal units that has been previously established (Schmitt, 2002). We emphasize that the only constraint on the parameters of the product units is that they yield real-valued, that is, not complex-valued, functions. This means further that the statement holds for networks of arbitrary order, that is, it does not impose any restrictions on the magnitude of the weights of the product units.

**Proposition 1.** (Schmitt, 2002, Theorem 2) *Suppose $\mathcal{N}$ is a feedforward neural network consisting of sigmoidal, product, and linear units. Let $k$ be its size and $W$ the number of adjustable weights. The VC dimension of $\mathcal{N}$ restricted to real-valued functions is at most $4(Wk)^2 + 20Wk \log(36Wk)$.*

## 3 Bounds on Computing Time

We establish bounds on the time required by recurrent neural networks for computing a family of functions $f_l : \mathbb{R} \to \mathbb{R}, l \geq 1$, where $l$ can be considered as a measure of the complexity of $f_l$. Specifically, $f_l$ is defined in terms of a dynamical system as the $l$th iterate of the logistic map $\phi(x) = 4x(1 - x)$, that is,

$$f_l(x) \;=\; \begin{cases} \phi(x) & l = 1, \\ \phi(f_{l-1}(x)) & l \geq 2. \end{cases}$$

We observe that there is a single recurrent network capable of computing every $f_l$ in time $O(l)$.

**Lemma 2.** *There is a general recurrent neural network that computes $f_l$ in time $2l + 1$ for every $l$.*

*Proof.* The network is shown in Figure 2. It consists of linear and second-order units. All computation nodes are initialized with 0, except $y_1$, which starts with 1 and outputs 0 during all following steps. The purpose of $y_1$ is to let the input $x$

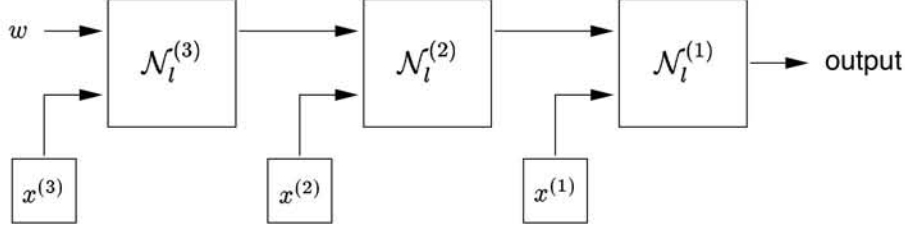

Figure 3: Network $\mathcal{N}_l$.

enter node $y_2$ at time 1 and keep it away at later times. Clearly, the value $f_l(x)$ results at node $y_5$ after $2l + 1$ steps. □

The network used for computing $f_l$ requires only linear and second-order units. The following result shows that the established upper bound is asymptotically almost tight, with a gap only of order four. Moreover, the lower bound holds for networks of unrestricted order and with sigmoidal units.

**Theorem 3.** *Every general recurrent neural network of size $k$ requires at least time $cl^{1/4}/k$ to compute function $f_l$, where $c > 0$ is some constant.*

*Proof.* The idea is to construct higher-order networks $\mathcal{N}_l$ of small size that have comparatively large VC dimension. Such a network will consist of linear and product units and hypothetical units that compute functions $f_j$ for certain values of $j$. We shall derive a lower bound on the VC dimension of these networks. Assuming that the hypothetical units can be replaced by time-bounded general recurrent networks, we determine an upper bound on the VC dimension of the resulting networks in terms of size and computing time using an idea from Koiran and Sontag (1998) and Proposition 1. The comparison of the lower and upper VC dimension bounds will give an estimate of the time required for computing $f_l$.

Network $\mathcal{N}_l$, shown in Figure 3, is a feedforward network composed of three networks $\mathcal{N}_l^{(1)}, \mathcal{N}_l^{(2)}, \mathcal{N}_l^{(3)}$. Each network $\mathcal{N}_l^{(\mu)}, \mu = 1, 2, 3$, has $l$ input nodes $x_1^{(\mu)}, \ldots, x_l^{(\mu)}$ and $2l + 2$ computation nodes $y_0^{(\mu)}, \ldots, y_{2l+1}^{(\mu)}$ (see Figure 4). There is only one adjustable parameter in $\mathcal{N}_l$, denoted $w$, all other weights are fixed. The computation nodes are defined as follows (omitting time parameter $t$):

$$y_0^{(\mu)} = \begin{cases} w & \text{for } \mu = 3, \\ y_{2l+1}^{(\mu+1)} & \text{for } \mu = 1, 2, \end{cases}$$

$$y_i^{(\mu)} = f_{l^{\mu-1}}(y_{i-1}^{(\mu)}) \text{ for } i = 1, \ldots, l \text{ and } \mu = 1, 2, 3,$$

$$y_{l+i}^{(\mu)} = y_i^{(\mu)} \cdot x_i^{(\mu)}, \text{ for } i = 1, \ldots, l \text{ and } \mu = 1, 2, 3,$$

$$y_{2l+1}^{(\mu)} = y_{l+1}^{(\mu)} + \cdots + y_{2l}^{(\mu)} \text{ for } \mu = 1, 2, 3.$$

The nodes $y_0^{(\mu)}$ can be considered as additional input nodes for $\mathcal{N}_l^{(\mu)}$, where $\mathcal{N}_l^{(3)}$ gets this input from $w$, and $\mathcal{N}_l^{(\mu)}$ from $\mathcal{N}_l^{(\mu+1)}$ for $\mu = 1, 2$. Node $y_{2l+1}^{(\mu)}$ is the output node of $\mathcal{N}_l^{(\mu)}$, and node $y_{2l+1}^{(1)}$ is also the output node of $\mathcal{N}_l$. Thus, the entire network has $3l + 6$ nodes that are linear or product units and $3l$ nodes that compute functions $f_1, f_l,$ or $f_{l^2}$.

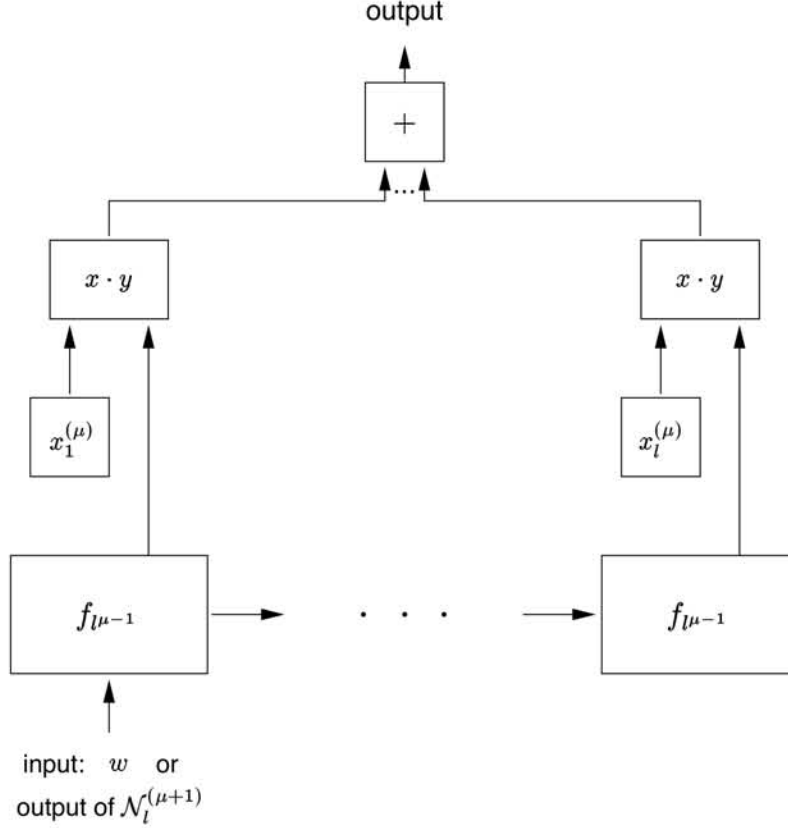

Figure 4: Network $\mathcal{N}_l^{(\mu)}$.

We show that $\mathcal{N}_l$ shatters some set of cardinality $l^3$, in particular, the set $S = (\{e_i : i = 1, \ldots, l\})^3$, where $e_i \in \{0, 1\}^l$ is the unit vector with a 1 in position $i$ and 0 elsewhere. Every dichotomy of $S$ can be programmed into the network parameter $w$ using the following fact about the logistic function $\phi$ (see Koiran and Sontag, 1998, Lemma 2): For every binary vector $b \in \{0, 1\}^m, b = b_1 \ldots b_m$, there is some real number $w \in [0, 1]$ such that for $i = 1, \ldots, m$

$$\phi^i(w) \quad \in \quad \begin{cases} [0, 1/2) & \text{if } b_i = 0, \\ (1/2, 1] & \text{if } b_i = 1. \end{cases}$$

Hence, for every dichotomy $(S_0, S_1)$ of $S$ the parameter $w$ can be chosen such that every $(e_{i_1}, e_{i_2}, e_{i_3}) \in S$ satisfies

$$\phi^{i_1}(\phi^{i_2 \cdot l}(\phi^{i_3 \cdot l^2}(w))) \quad \begin{cases} < & 1/2 \quad \text{if } (e_{i_1}, e_{i_2}, e_{i_3}) \in S_0, \\ > & 1/2 \quad \text{if } (e_{i_1}, e_{i_2}, e_{i_3}) \in S_1. \end{cases}$$

Since $f_{i_1 + i_2 \cdot l + i_3 \cdot l^2}(w) = \phi^{i_1}(\phi^{i_2 \cdot l}(\phi^{i_3 \cdot l^2}(w)))$, this is the value computed by $\mathcal{N}_l$ on input $(e_{i_1}, e_{i_2}, e_{i_3})$, where $e_{i_\mu}$ is the input given to network $\mathcal{N}_l^{(\mu)}$. (Input $e_{i_\mu}$ selects the function $f_{i_\mu \cdot l^{\mu-1}}$ in $\mathcal{N}_l^{(\mu)}$.) Hence, $S$ is shattered by $\mathcal{N}_l$, implying that $\mathcal{N}_l$ has VC dimension at least $l^3$.

Assume now that $f_j$ can be computed by a general recurrent neural network of size at most $k_j$ in time $t_j$. Using an idea of Koiran and Sontag (1998), we unfold the network to obtain a feedforward network of size at most $k_j t_j$ computing $f_j$. Thus we can replace the nodes computing $f_1, f_l, f_{l^2}$ in $\mathcal{N}_l$ by networks of size $k_1 t_1, k_l t_l, k_{l^2} t_{l^2}$, respectively, such that we have a feedforward network $\mathcal{N}'_l$ consisting of sigmoidal, product, and linear units. Since there are $3l$ units in $\mathcal{N}_l$ computing $f_1, f_l$, or $f_{l^2}$ and at most $3l + 6$ product and linear units, the size of $\mathcal{N}'_l$ is at most $c_1 l k_{l^2} t_{l^2}$ for some constant $c_1 > 0$. Using that $\mathcal{N}'_l$ has one adjustable weight, we get from Proposition 1 that its VC dimension is at most $c_2 l^2 k_{l^2}^2 t_{l^2}^2$ for some constant $c_2 > 0$. On the other hand, since $\mathcal{N}_l$ and $\mathcal{N}'_l$ both shatter $S$, the VC dimension of $\mathcal{N}'_l$ is at least $l^3$. Hence, $l^3 \leq c_2 l^2 k_{l^2}^2 t_{l^2}^2$ holds, which implies that $t_{l^2} \geq c l^{1/2}/k_{l^2}$ for some $c > 0$, and hence $t_l \geq c l^{1/4}/k_l$. $\qquad\square$

Lemma 2 shows that a single recurrent network is capable of computing every function $f_l$ in time $O(l)$. The following consequence of Theorem 3 establishes that this bound cannot be much improved.

**Corollary 4.** *Every general recurrent neural network requires at least time $\Omega(l^{1/4})$ to compute the functions $f_l$.*

# 4 Conclusions and Perspectives

We have established bounds on the computing time of analog recurrent neural networks. The result shows that for every network of given size there are functions of arbitrarily high time complexity. This fact does not rely on a bound on the magnitude of weights. We have derived upper and lower bounds that are rather tight—with a polynomial gap of order four—and hold for the computation of a specific family of real-valued functions in one variable. Interestingly, the upper bound is shown using second-order networks without sigmoidal units, whereas the lower bound is valid even for networks with sigmoidal units and arbitrary product units. This indicates that adding these units might decrease the computing time only marginally. The derivation made use of an upper bound on the VC dimension of higher-order sigmoidal networks. This bound is not known to be optimal. Any future improvement will therefore lead to a better lower bound on the computing time.

We have focussed on product and sigmoidal units as nonlinear computing elements. However, the construction presented here is generic. Thus, it is possible to derive similar results for radial basis function units, models of spiking neurons, and other unit types that are known to yield networks with bounded VC dimension. The questions whether such results can be obtained for continuous-time networks and for networks operating in the domain of complex numbers, are challenging. A further assumption made here is that the networks compute the functions exactly. By a more detailed analysis and using the fact that the shattering of sets requires the outputs only to lie below or above some threshold, similar results can be obtained for networks that approximate the functions more or less closely and for networks that are subject to noise.

### Acknowledgment

The author gratefully acknowledges funding from the Deutsche Forschungsgemeinschaft (DFG). This work was also supported in part by the ESPRIT Working Group in Neural and Computational Learning II, NeuroCOLT2, No. 27150.

# References

Anthony, M. and Bartlett, P. L. (1999). *Neural Network Learning: Theoretical Foundations*. Cambridge University Press, Cambridge.

Balcázar, J., Gavaldà, R., and Siegelmann, H. T. (1997). Computational power of neural networks: A characterization in terms of Kolmogorov complexity. *IEEE Transcations on Information Theory*, 43:1175–1183.

Blum, L., Cucker, F., Shub, M., and Smale, S. (1998). *Complexity and Real Computation*. Springer-Verlag, New York.

Carrasco, R. C., Forcada, M. L., Valdés-Muñoz, M. A., and Ñeco, R. P. (2000). Stable encoding of finite state machines in discrete-time recurrent neural nets with sigmoid units. *Neural Computation*, 12:2129–2174.

Durbin, R. and Rumelhart, D. (1989). Product units: A computationally powerful and biologically plausible extension to backpropagation networks. *Neural Computation*, 1:133–142.

Gavaldà, R. and Siegelmann, H. T. (1999). Discontinuities in recurrent neural networks. *Neural Computation*, 11:715–745.

Haykin, S. (1999). *Neural Networks: A Comprehensive Foundation*. Prentice Hall, Upper Saddle River, NJ, second edition.

Kilian, J. and Siegelmann, H. T. (1996). The dynamic universality of sigmoidal neural networks. *Information and Computation*, 128:48–56.

Koiran, P. and Sontag, E. D. (1998). Vapnik-Chervonenkis dimension of recurrent neural networks. *Discrete Applied Mathematics*, 86:63–79.

Maass, W., Natschläger, T., and Markram, H. (2001). Real-time computing without stable states: A new framework for neural computation based on perturbations. Preprint.

Maass, W. and Orponen, P. (1998). On the effect of analog noise in discrete-time analog computations. *Neural Computation*, 10:1071–1095.

Maass, W. and Sontag, E. D. (1999). Analog neural nets with Gaussian or other common noise distributions cannot recognize arbitrary regular languages. *Neural Computation*, 11:771–782.

Omlin, C. W. and Giles, C. L. (1996). Constructing deterministic finite-state automata in recurrent neural networks. *Journal of the Association for Computing Machinery*, 43:937–972.

Schmitt, M. (2002). On the complexity of computing and learning with multiplicative neural networks. *Neural Computation*, 14. In press.

Siegelmann, H. T. (1999). *Neural Networks and Analog Computation: Beyond the Turing Limit*. Progress in Theoretical Computer Science. Birkhäuser, Boston.

Siegelmann, H. T. and Sontag, E. D. (1995). On the computational power of neural nets. *Journal of Computer and System Sciences*, 50:132–150.

Šíma, J. and Orponen, P. (2001). Exponential transients in continuous-time symmetric Hopfield nets. In Dorffner, G., Bischof, H., and Hornik, K., editors, *Proceedings of the International Conference on Artificial Neural Networks ICANN 2001*, volume 2130 of *Lecture Notes in Computer Science*, pages 806–813, Springer, Berlin.
